# Schema Learning: Experience-Based Construction of Predictive Action Models

**Michael P. Holmes**
College of Computing
Georgia Institute of Technology
Atlanta, GA 30332-0280
mph@cc.gatech.edu

**Charles Lee Isbell, Jr.**
College of Computing
Georgia Institute of Technology
Atlanta, GA 30332-0280
isbell@cc.gatech.edu

## Abstract

Schema learning is a way to discover probabilistic, constructivist, predictive action models (schemas) from experience. It includes methods for finding and using hidden state to make predictions more accurate. We extend the original schema mechanism [1] to handle arbitrary discrete-valued sensors, improve the original learning criteria to handle POMDP domains, and better maintain hidden state by using schema predictions. These extensions show large improvement over the original schema mechanism in several rewardless POMDPs, and achieve very low prediction error in a difficult speech modeling task. Further, we compare extended schema learning to the recently introduced predictive state representations [2], and find their predictions of next-step action effects to be approximately equal in accuracy. This work lays the foundation for a schema-based system of integrated learning and planning.

## 1 Introduction

*Schema learning*[1] is a data-driven, constructivist approach for discovering probabilistic action models in dynamic controlled systems. Schemas, as described by Drescher [1], are probabilistic units of cause and effect reminiscent of STRIPS operators [3]. A schema predicts how specific sensor values will change as different actions are executed from within particular sensory contexts. The learning mechanism also discovers hidden state features in order to make schema predictions more accurate.

In this work we have generalized and extended Drescher's original mechanism to learn more accurate predictions by using improved criteria both for discovery and refinement of schemas as well as for creation and maintenance of hidden state. While Drescher's work included mechanisms for action selection, here we focus exclusively on the problem of learning schemas and hidden state to accurately model the world. In several benchmark POMDPs, we show that our extended schema learner produces significantly better action models than the original. We also show that the extended learner performs well on a complex, noisy speech modeling task, and that its prediction accuracy is approximately equal to that of predictive state representations [2] on a set of POMDPs, with faster convergence.

## 2  Schema Learning

Schema learning is a process of constructing probabilistic action models of the environment so that the effects of agent actions can be predicted. Formally, a schema learner is fitted with a set of sensors $S = \{s_1, s_2, \ldots\}$ and a set of actions $A = \{a_1, a_2, \ldots\}$ through which it can perceive and manipulate the environment. Sensor values are discrete: $s_i^j$ means that $s_i$ has value $j$. As it observes the effects of its actions on the environment, the learner constructs predictive units of sensorimotor cause and effect called *schemas*. A schema $C \xrightarrow{a_i} R$ essentially says, "If I take action $a_i$ in situation $C$, I will see result $R$." Schemas thus have three components: (1) the context $C = \{c_1, c_2, \ldots, c_n\}$, which is a set of sensor conditions $c_i \equiv s_j^k$ that must hold for the schema to be applicable, (2) the action that is taken, and (3) the result, which is a set of sensor conditions $R = \{r_1, r_2, \ldots, r_m\}$ predicted to follow the action. A schema is said to be *applicable* if its context conditions are satisfied, *activated* if it is applicable and its action is taken, and to *succeed* if it is activated and the predicted result is observed. Schema quality is measured by *reliability*, which is the probability that activation culminates in success: $Rel(C \xrightarrow{a_i} R) = prob(R_{t+1}|C_t, a_{i(t)})$.

Note that schemas are not rules telling an agent what to do; rather, they are descriptions of what will happen if the agent takes a particular action in a specific circumstance. Also note that schema learning has no predefined states such as those found in a POMDP or HMM; the set of sensor readings *is* the state. Because one schema's result can set up another schema's context, schemas fit naturally into a planning paradigm in which they are chained from the current situation to reach sensor-defined goals.

### 2.1  Discovery and Refinement

Schema learning comprises two basic phases: *discovery*, in which context-free action/result schemas are found, and *refinement*, in which context is added to increase reliability. In discovery, statistics track the influence of each action $a_i$ on each sensor condition $s_r^j$. Drescher's original schema mechanism accommodated only binary-valued sensors, but we have generalized it to allow a heterogeneous set of sensors that take on arbitrary discrete values. In the present work, we assume that the effects of actions are observed on the subsequent timestep, which leads to the following criterion for discovering action effects:

$$count(a_t, s_{r(t+1)}^j) > \theta_d, \tag{1}$$

where $\theta_d$ is a noise-filtering threshold. If this criterion is met, the learner constructs a schema $\emptyset \xrightarrow{a_i} s_r^j$, where the empty set, $\emptyset$, means that the schema is applicable in any situation. This works in a POMDP because it means that executing $a_i$ in some state has caused sensor $s_r$ to give observation $j$, implying that such a transition exists in the underlying (but unknown) system model. The presumption is that we can later learn what sensory context makes this transition reliable. Drescher's original discovery criterion generalizes in the non-binary case to:

$$\frac{prob(s_{r(t+1)}^j|a_t)}{prob(s_{r(t+1)}^j|\overline{a}_t)} > \theta_{od}, \tag{2}$$

where $\theta_{od} > 1$ and $\overline{a}_t$ means $a$ was not taken at time $t$. Experiments in worlds of known structure show that this criterion misses many true action effects.

When a schema is discovered, it has no context, so its reliability may be low if the effect occurs only in particular situations. Schemas therefore begin to look for context conditions

| Criterion | Extended Schema Learner | Original Schema Learner |
|---|---|---|
| Discovery | $count(a_t, s^j_{r(t+1)}) > \theta_d$ | $\dfrac{prob(s^j_{r(t+1)}\|a_t)}{prob(s^j_{r(t+1)}\|\bar{a}_t)} > \theta_{od}$<br>Binary sensors only |
| Refinement | $\dfrac{Rel(C \cup \{s^j_c\} \xrightarrow{a_i} R)}{Rel(C \xrightarrow{a_i} R)} > \theta$<br>Annealed threshold | $\dfrac{Rel(C \cup \{s^j_c\} \xrightarrow{a_i} R)}{Rel(C \xrightarrow{a_i} R)} > \theta$<br>Static threshold<br>Binary sensors only |
| Synthetic Item Creation | $0 < Rel(C \xrightarrow{a_i} R) < \theta$<br>No context refinement possible | $0 < Rel(C \xrightarrow{a_i} R) < \theta$<br>Schema is locally consistent |
| Synthetic Item Maintenance | Predicted by other schemas | Average duration |

Table 1: **Comparison of extended and original schema learners.**

that increase reliability. The criterion for adding $s^j_c$ to the context of $C \xrightarrow{a_i} R$ is:

$$\frac{Rel(C \cup \{s^j_c\} \xrightarrow{a_i} R)}{Rel(C \xrightarrow{a_i} R)} > \theta_c, \tag{3}$$

where $\theta_c > 1$. In practice we have found it necessary to anneal $\theta_c$ to avoid adding spurious context. Once the criterion is met, a child schema $C' \xrightarrow{a_i} R$ is formed, where $C' = C \cup s^j_c$.

## 2.2 Synthetic Items

In addition to basic discovery and refinement of schemas, a schema learner also discovers hidden state. Consider the case where no context conditions are found to make a schema reliable. There must be unperceived environmental factors on which the schema's reliability depends (see [4]). The schema learner therefore creates a new binary-valued virtual sensor, called a *synthetic item*, to represent the presence of conditions in the environment that allow the schema to succeed. This addresses the state aliasing problem by splitting the state space into two parts, one where the schema succeeds, and one where it does not. Synthetic items are said to *reify* the *host schemas* whose success conditions they represent; they have value 1 if the host schema would succeed if activated, and value 0 otherwise. Upon creation, a synthetic item begins to act as a normal sensor, with one exception: the agent has no way of directly perceiving its value. Creation and state maintenance criteria thus emerge as the main problems associated with synthetic items.

Drescher originally posited two conditions for the creation of a synthetic item: (1) a schema must be unreliable, and (2) the schema must be locally consistent, meaning that if it succeeds once, it has a high probability of succeeding again if activated soon afterward. The second of these conditions formalizes the assumption that a well-behaved environment has persistence and does not tend to radically change from moment to moment. This was motivated by the desire to capture Piagetian "conservation phenomena." While well-motivated, we have found that the second condition is simply too restrictive. Our criterion for creating synthetic items is $0 < Rel(C \xrightarrow{a_i} R) < \theta_r$, subject to the constraint that the statistics governing possible additional context conditions have converged. When this criterion is met, a synthetic item is created and is thenceforth treated as a normal sensor, able to be incorporated into the contexts and results of other schemas.

A newly created synthetic item is grounded: it represents whatever conditions in the world allow the host schema to succeed when activated. Thus, upon activation of the host schema, we *retroactively* know the state of the synthetic item at the time of activation (1 if the schema succeeded, 0 otherwise). Because the synthetic item is treated as a sensor, we can

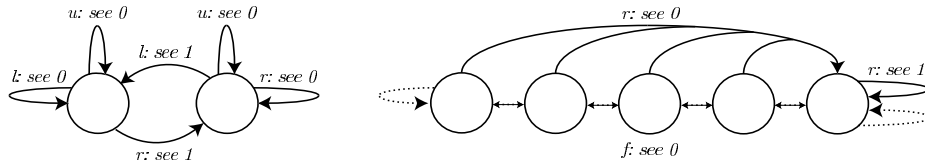

Figure 1: **Benchmark problems.** (left) The flip system. All transitions are deterministic. (right) The float/reset system. Dashed lines represent float transitions that happen with probability 0.5, while solid lines represent deterministic reset transitions.

discover which previous actions led to each synthetic item state, and the synthetic item can come to be included as a result condition in new schemas. Once we have reliable schemas that predict the state of a synthetic item, we can begin to know its state non-retroactively, without having to activate the host schema. The synthetic item's state can potentially be known just as well as that of the regular sensors, and its addition expands the state representation in just such a way as to make sensory predictions more reliable. Predicted synthetic item state implicitly summarizes the relevant preceding history: it indicates that one of the schemas that predicts it was just activated. If the predicting schema also has a synthetic item in its context, an additional step of history is implied. Such chaining allows synthetic items to summarize arbitrary amounts of history without explicitly remembering any of it. This use of schemas to predict synthetic item state is in contrast to [1], which relied on the average duration of synthetic item states in order to predict them. Table 1 compares our extended schema learning criteria with Drescher's original criteria.

## 3 Empirical Evaluation

In order to test the advantages of the extended learning criteria, we compared four versions of schema learning. The first two were basic learners that made no use of synthetic items, but discovered and refined schemas using our extended criteria in one case, and the direct generalizations of Drescher's original criteria in the other. The second pair added the extended and original synthetic item mechanisms, respectively, to the first pair.

Our first experimental domains are based on those used in [5]. They have a mixture of transient and persistent hidden state and, though small, are non-trivial.[2] The flip system is shown on the left in Figure 1; it features deterministic transitions, hidden state, and a null action that confounds simplistic history approaches to handling hidden state. The float/reset system is illustrated on the right side of Figure 1; it features both deterministic and stochastic transitions, as well as a more complicated hidden state structure. Finally, we use a modified float/reset system in which the $f$ action from the two right-most states leads deterministically to their left neighbor; this reveals more about the hidden state structure.

To test predictive power, each schema learner, upon taking an action, uses the most reliable of all activated schemas to predict what the next value of each sensor will be. If there is no activation of a reliable schema to predict the value of a particular sensor, its value is predicted to stay constant. Error is measured as the fraction of incorrect predictions.

In these experiments, actions were chosen uniformly at random, and learning was allowed to continue throughout.[3] No learning parameters are changed over time; schemas stop being created when discovery and refinement criteria cease to generate them. Figure 2 shows the performance in each domain, while Table 2 summarizes the average error.

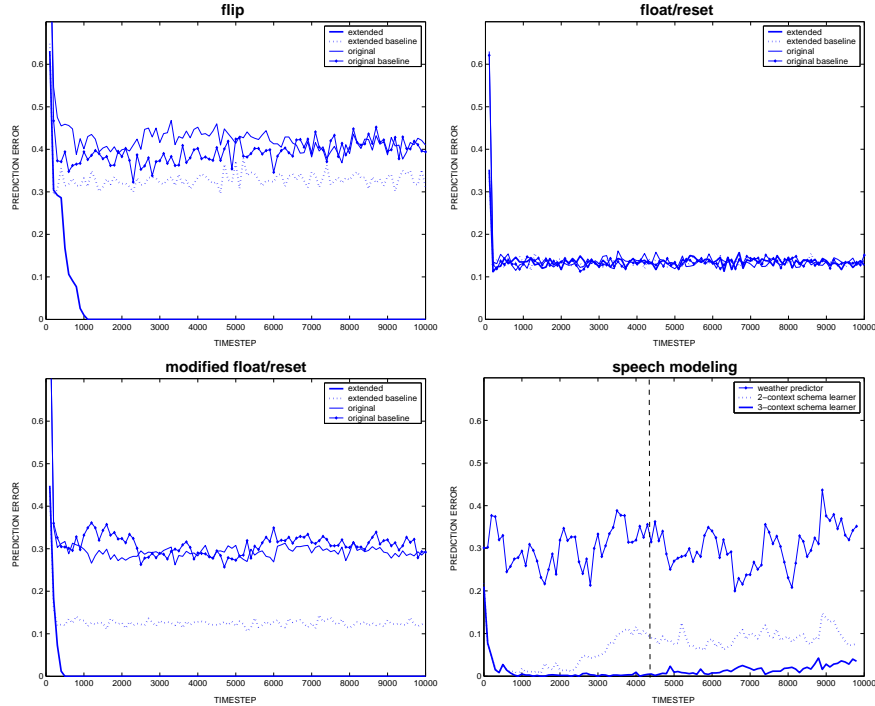

Figure 2: **Prediction error in several domains.** The $x$-axis represents timesteps and the $y$-axis represents error. Each point represents average error over 100 timesteps. In the speech modeling graph, learning is stopped after approximately 4300 timesteps (shown by the vertical line), after which no schemas are added, though reliabilities continue to be updated.

| Learner | flip | float/reset | modified f/r |
|---|---|---|---|
| Extended | 0.020 | 0.136 | 0.00716 |
| Extended baseline | 0.331 | 0.136 | 0.128 |
| Original | 0.426 | 0.140 | 0.299 |
| Original baseline | 0.399 | 0.139 | 0.315 |

Table 2: **Average error.** Calculated over 10 independent runs of 10,000 timesteps each.

## 3.1 Speech Modeling

The Japanese vowel dataset [6] contains time-series recordings of nine Japanese speakers uttering the *ae* vowel combination 54-118 times. Each data point consists of 12 continuous-valued cepstral coefficients, which we transform into 12 sensors with five discrete values each. The data is noisy and the dynamics are non-stationary between speakers. Each utterance is divided in half, with the first half treated as the action of speaking *a* and the latter half as *e*. In order to more quickly adapt to discontinuity resulting from changes in speaker, reliability was calculated using an exponential weighting of more recent observations; each relevant probability $p$ was updated according to:

$$p_{t+1} = \alpha p_t + (1 - \alpha) \begin{cases} 1 \text{ if event occurred at time } t \\ 0 \text{ otherwise} \end{cases} . \quad (4)$$

The parameter $\alpha$ is set equal to the current prediction accuracy so that decreased accuracy causes faster adaptation. Several modifications were necessary for tractability: (1) schemas whose reliability fell below a threshold of their parents' reliability were removed, (2) con-

text sizes were, on separate experimental runs, restricted to two and three items, and (3) the synthetic item mechanisms were deactivated. Figure 2 displays results for this learner compared to a baseline weather predictor.[4]

## 3.2   Analysis

In each benchmark problem, the learners drop to minimum error after no more than 1000 timesteps. Large divergence in the curves corresponds to the creation of synthetic items and the discovery of schemas that predict synthetic item state. Small divergence corresponds to differences in discovery and refinement criteria. In flip and modified float/reset, the extended schema learner reaches zero error, having a complete model of the hidden state, and outperforms all other learners, while the extended basic version outperforms both original learners. In float/reset, all learners perform approximately equally, reflecting the fact that, given the hidden stochasticity of this system, the best schema for action $r$ is one that, without reference to synthetic items, gives a prediction of $1$. Surprisingly, the original learner never significantly outperformed its baseline, and even performed worse than the baseline in flip. This is accounted for by the duration-based maintenance of synthetic items, which causes the original learner to maintain transient synthetic item state longer than it should. Prediction-based synthetic item maintenance overcomes this limitation.

The speech modeling results show that schema learning can induce high-quality action models in a complex, noisy domain. With a maximum of three context conditions, it averaged only 1.2% error while learning, and 1.6% after learning stopped, a large improvement over the 30.3% error of the baseline weather predictor. Note that allowing three instead of two context conditions dropped the error from 4.6% to 1.2% and from 9.0% to 1.6% in the training and testing phases, respectively, demonstrating the importance of incremental specialization of schemas through context refinement.

All together, these results show that our extended schema learner produces better action models than the original, and can handle more complex domains. Synthetic items are seen to effectively model hidden state, and prediction-based maintenance of synthetic item state is shown to be more accurate than duration-based maintenance in POMDPs. Discovery of schemas is improved by our criterion, missing fewer legitimate schemas, and therefore producing more accurate predictions. Refinement using the annealed generalization of the original criterion performs correctly with a lower false positive rate.

## 4   Comparison to Predictive State Representations

Predictive state representations (PSRs; [2]), like schema learning, are based on grounded, sensorimotor predictions that uncover hidden state. Instead of schemas, PSRs rely on the notion of tests. A test $q$ is a series of alternating actions and observations $a_0 o_0 a_1 o_1 \ldots a_n o_n$. In a PSR, the environment state is represented as the probabilities that each of a set of core tests would yield its observations if its actions were executed. These probabilities are updated at each timestep by combining the current state with the new action/observation pair. In this way, the PSR implicitly contains a sufficient history-based statistic for prediction, and should overcome aliasing relative to immediate observations. [2] shows that linear PSRs are at least as compact and general as POMDPs, while [5] shows that PSRs can learn to accurately maintain their state in several POMDP problems.

A schema is similar to a one-step PSR test, and schema reliability roughly corresponds to the probability of a PSR test. Schemas differ, however, in that they only specify context and result incrementally, incorporating incremental history via synthetic items, while PSR tests incorporate the complete history and full observations (i.e. all sensor readings at once) into

| Problem | PSR | Schema Learner | Difference | Schema Learning Steps |
|---|---|---|---|---|
| flip | 0 | 0 | 0 | 10,000 |
| float/reset | 0.11496 | 0.13369 | 0.01873 | 10,000 |
| network | 0.04693 | 0.06457 | 0.01764 | 10,000 |
| paint | 0.20152 | 0.21051 | 0.00899 | 30,000 |

Table 3: **Prediction error for PSRs and schema learning on several POMDPs.** Error is averaged over 10 epochs of 10,000 timesteps each. Performance differs by less than 2% in every case.

a test probability. A multi-step test can say more about the current state than a schema, but is not as useful for regression planning because there is no way to extract the probability that a particular one of its observations will be obtained. Thus, PSRs are more useful as Markovian state for reinforcement learning, while schemas are useful for explicit planning. Note that synthetic items and PSR core test probabilities both attempt to capture a sufficient history statistic without explicitly maintaining history. This suggests a deeper connection between the two approaches, but the relationship has yet to be formalized.

We compared the predictive performance of PSRs with that of schema learning on some of the POMDPs from [5]. One-step PSR core tests can be used to predict observations: as an action is taken, the probability of each observation is the probability of the one-step core test that uses the current action and terminates in that observation. We choose the most probable observation as the PSR prediction. This allows us to evaluate PSR predictions using the same error measure (fraction of incorrect predictions) as in schema learning.[5]

In our experiments, the extended schema learner was first allowed to learn until it reached an asymptotic minimum error (no longer than 30,000 steps). Learning was then deactivated, and the schema learner and PSR each made predictions over a series of randomly chosen actions. Table 3 presents the average performance for each approach.

Learning PSR parameters required 1-10 million timesteps [5], while schema learning used no more than 30,000 steps. Also, learning PSR parameters required access to the underlying POMDP [5], whereas schema learning relies solely on sensorimotor information.

## 5   Related Work

Aside from PSRs, schema learning is also similar to older work in learning planning operators, most notably that of Wang [7], Gil [8], and Shen [9]. These approaches use observations to learn classical, deterministic STRIPS-like operators in predicate logic environments. Unlike schema learning, they make the strong assumption that the environment does not produce noisy observations. Wang and Gil further assume no perceptual aliasing.

Other work in this area has attempted to handle noise, but only in the problem of context refinement. Benson [10] gives his learner prior knowledge about action effects, and the learner finds conditions to make the effects reliable with some tolerance for noise. One advantage of Benson's formalism is that his operators are durational, rather than atomic over a single timestep. Balac et al. [11] use regression trees to find regions of noisy, continuous sensor space that cause a specified action to vary in the degree of its effect.

Finally, Shen [9] and McCallum [12] have mechanisms for handling state aliasing. Shen uses differences in successful and failed predictions to identify pieces of history that reveal hidden state. His approach, however, is completely noise intolerant. McCallum's UTree algorithm selectively adds pieces of history in order to maximize prediction of reward.

This bears a strong resemblance to the history represented by chains of synthetic items, a connection that should be explored more fully. Synthetic items, however, are for general sensor prediction, which contrasts with UTree's task-specific focus on reward prediction. Schema learning, PSRs, and the UTree algorithm are all highly related in this sense of selectively tracking history information to improve predictive performance.

## 6 Discussion and Future Work

We have shown that our extended schema learner produces accurate action models for a variety of POMDP systems and for a complex speech modeling task. The extended schema learner performs substantially better than the original, and compares favorably in predictive power to PSRs while appearing to learn much faster. Building probabilistic goal-regression planning on top of the schemas is a logical next step; however, to succeed with real-world planning problems, we believe that we need to extend the learning mechanism in several ways. For example, the schema learner must explicitly handle actions whose effects occur over an extended duration instead of after one timestep. The learner should also be able to directly handle continuous-valued sensors. Finally, the current mechanism has no means of abstracting similar schemas, e.g., to reduce $x_1^1 \xrightarrow{a} x_1^2$ and $x_1^2 \xrightarrow{a} x_1^3$ to $x_1^p \xrightarrow{a} x_1^{p+1}$.

**Acknowledgements**

Thanks to Satinder Singh and Michael R. James for providing POMDP PSR parameters.

## Footnotes

[1]This use of the term *schema* derives from Piaget's usage in the 1950s; it bears no relation to database schemas or other uses of the term.

[2]E.g. [5] showed that flip is non-trivial because it cannot be modeled exactly by k-Markov models, and its EM-trained POMDP representations require far more than the minimum number of states.

[3]Note that because a prediction is made before each observation, the observation does not contribute to the learning upon which its predicted value is based.

[4]A weather predictor always predicts that values will stay the same as they are presently.

[5]Unfortunately, not all the POMDPs from [5] had one-step core tests to cover the probability of every observation given every action. We restricted our comparisons to the four systems that had at least two actions for which the probability of all next-step observations could be determined.

## References

[1] G. Drescher. *Made-up minds: a constructivist approach to artificial intelligence*. MIT Press, 1991.

[2] M. L. Littman, R. S. Sutton, and S. Singh. Predictive representations of state. In *Advances in Neural Information Processing Systems*, pages 1555–1561. MIT Press, 2002.

[3] R. E. Fikes and N. J. Nilsson. STRIPS: a new approach to the application of theorem proving to problem solving. *Artificial Intelligence*, 2:189–208, 1971.

[4] C. T. Morrison, T. Oates, and G. King. Grounding the unobservable in the observable: the role and representation of hidden state in concept formation and refinement. In *AAAI Spring Symposium on Learning Grounded Representations*, pages 45–49. AAAI Press, 2001.

[5] S. Singh, M. L. Littman, N. K. Jong, D. Pardoe, and P. Stone. Learning predictive state representations. In *International Conference on Machine Learning*, pages 712–719. AAAI Press, 2003.

[6] M. Kudo, J. Toyama, and M. Shimbo. Multidimensional curve classification using passing-through regions. *Pattern Recognition Letters*, 20(11–13):1103–1111, 1999.

[7] X. Wang. Learning by observation and practice: An incremental approach for planning operator acquisition. In *International Conference on Machine Learning*, pages 549–557. AAAI Press, 1995.

[8] Y. Gil. Learning by experimentation: Incremental refinement of incomplete planning domains. In *International Conference on Machine Learning*, pages 87–95. AAAI Press, 1994.

[9] W.-M. Shen. Discovery as autonomous learning from the environment. *Machine Learning*, 12:143–165, 1993.

[10] Scott Benson. Inductive learning of reactive action models. In *International Conference on Machine Learning*, pages 47–54. AAAI Press, 1995.

[11] N. Balac, D. M. Gaines, and D. Fisher. Using regression trees to learn action models. In *IEEE Systems, Man and Cybernetics Conference*, 2000.

[12] A. W. McCallum. *Reinforcement Learning with Selective Perception and Hidden State*. PhD thesis, University of Rochester, 1995.
